# Markov processes on curves for automatic speech recognition

**Lawrence Saul and Mazin Rahim**
AT&T Labs — Research
Shannon Laboratory
180 Park Ave E-171
Florham Park, NJ 07932
{lsaul,mazin}@research.att.com

## Abstract

We investigate a probabilistic framework for automatic speech recognition based on the intrinsic geometric properties of curves. In particular, we analyze the setting in which two variables—one continuous ($x$), one discrete ($s$)—evolve jointly in time. We suppose that the vector $x$ traces out a smooth multidimensional curve and that the variable $s$ evolves stochastically as a function of the *arc length* traversed along this curve. Since arc length does not depend on the rate at which a curve is traversed, this gives rise to a family of Markov processes whose predictions, $\Pr[s|x]$, are invariant to nonlinear warpings of time. We describe the use of such models, known as *Markov processes on curves* (MPCs), for automatic speech recognition, where $x$ are acoustic feature trajectories and $s$ are phonetic transcriptions. On two tasks—recognizing New Jersey town names and connected alpha-digits—we find that MPCs yield lower word error rates than comparably trained hidden Markov models.

## 1  Introduction

Variations in speaking rate currently present a serious challenge for automatic speech recognition (ASR) (Siegler & Stern, 1995). It is widely observed, for example, that fast speech is more prone to recognition errors than slow speech. A related effect, occurring at the phoneme level, is that consonants are more frequently botched than vowels. Generally speaking, consonants have short-lived, non-stationary acoustic signatures; vowels, just the opposite. Thus, at the phoneme level, we can view the increased confusability of consonants as a consequence of *locally* fast speech.

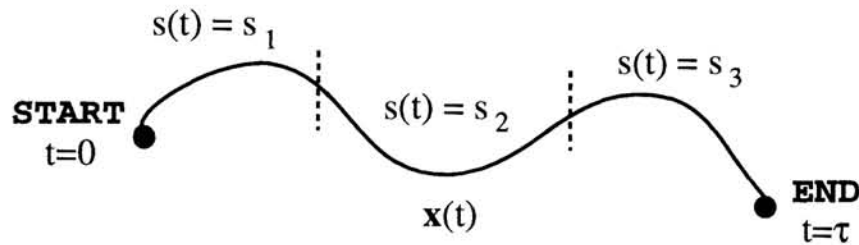

Figure 1: Two variables—one continuous ($x$), one discrete ($s$)—evolve jointly in time. The trace of $s$ partitions the curve of $x$ into different segments whose boundaries occur where $s$ changes value.

In this paper, we investigate a probabilistic framework for ASR that models variations in speaking rate as arising from *nonlinear warpings of time* (Tishby, 1990). Our framework is based on the observation that acoustic feature vectors trace out continuous trajectories (Ostendorf et al, 1996). We view these trajectories as multidimensional curves whose intrinsic geometric properties (such as arc length or radius) do not depend on the rate at which they are traversed (do Carmo, 1976). We describe a probabilistic model whose predictions are based on these intrinsic geometric properties and—as such—are invariant to nonlinear warpings of time. The handling of this invariance distinguishes our methods from traditional hidden Markov models (HMMs) (Rabiner & Juang, 1993).

The probabilistic models studied in this paper are known as *Markov processes on curves* (MPCs). The theoretical framework for MPCs was introduced in an earlier paper (Saul, 1997), which also discussed the problems of decoding and parameter estimation. In the present work, we report the first *experimental* results for MPCs on two difficult benchmark problems in ASR. On these problems—recognizing New Jersey town names and connected alpha-digits—our results show that MPCs generally match or exceed the performance of comparably trained HMMs.

The organization of this paper is as follows. In section 2, we review the basic elements of MPCs and discuss important differences between MPCs and HMMs. In section 3, we present our experimental results and evaluate their significance.

## 2   Markov processes on curves

Speech recognizers take a continuous acoustic signal as input and return a sequence of discrete labels representing phonemes, syllables, or words as output. Typically the short-time properties of the speech signal are summarized by acoustic feature vectors. Thus the abstract mathematical problem is to describe a multidimensional trajectory $\{x(t)|t \in [0, \tau]\}$ by a sequence of discrete labels $s_1 s_2 \ldots s_n$. As shown in figure 1, this is done by specifying consecutive time intervals such that $s(t) = s_k$ for $t \in [t_{k-1}, t_k]$ and attaching the labels $s_k$ to contiguous arcs along the trajectory. To formulate a probabilistic model of this process, we consider two variables—one continuous ($x$), one discrete ($s$)—that evolve jointly in time. Thus the vector $x$ traces out a smooth multidimensional curve, to each point of which the variable $s$ attaches a discrete label.

Markov processes on curves are based on the concept of *arc length*. After reviewing how to compute arc lengths along curves, we introduce a family of Markov processes whose predictions are invariant to nonlinear warpings of time. We then consider the ways in which these processes (and various generalizations) differ from HMMs.

## 2.1 Arc length

Let $g(\boldsymbol{x})$ define a $D \times D$ matrix-valued function over $\boldsymbol{x} \in \mathcal{R}^D$. If $g(\boldsymbol{x})$ is everywhere non-negative definite, then we can use it as a *metric* to compute distances along curves. In particular, consider two nearby points separated by the infinitesimal vector $d\boldsymbol{x}$. We define the squared distance between these two points as:

$$d\ell^2 = d\boldsymbol{x}^T g(\boldsymbol{x}) \, d\boldsymbol{x}. \tag{1}$$

Arc length along a curve is the non-decreasing function computed by integrating these local distances. Thus, for the trajectory $\boldsymbol{x}(t)$, the arc length between the points $\boldsymbol{x}(t_1)$ and $\boldsymbol{x}(t_2)$ is given by:

$$\ell = \int_{t_1}^{t_2} dt \, \left[ \dot{\boldsymbol{x}}^T g(\boldsymbol{x}) \, \dot{\boldsymbol{x}} \right]^{\frac{1}{2}}, \tag{2}$$

where $\dot{\boldsymbol{x}} = \frac{d}{dt}[\boldsymbol{x}(t)]$ denotes the time derivative of $\boldsymbol{x}$. Note that the arc length defined by eq. (2) is invariant under reparameterizations of the trajectory, $\boldsymbol{x}(t) \to \boldsymbol{x}(f(t))$, where $f(t)$ is any smooth monotonic function of time that maps the interval $[t_1, t_2]$ into itself.

In the special case where $g(\boldsymbol{x})$ is the identity matrix, eq. (2) reduces to the standard definition of arc length in Euclidean space. More generally, however, eq. (1) defines a non-Euclidean metric for computing arc lengths. Thus, for example, if the metric $g(\boldsymbol{x})$ varies as a function of $\boldsymbol{x}$, then eq. (2) can assign different arc lengths to the trajectories $\boldsymbol{x}(t)$ and $\boldsymbol{x}(t) + \boldsymbol{x}_0$, where $\boldsymbol{x}_0$ is a constant displacement.

## 2.2 States and lifelengths

We now return to the problem of segmentation, as illustrated in figure 1. We refer to the possible values of $s$ as *states*. MPCs are conditional random processes that evolve the state variable $s$ stochastically as a function of the arc length traversed along the curve of $\boldsymbol{x}$. In MPCs, *the probability of remaining in a particular state decays exponentially with the cumulative arc length traversed in that state*. The signature of a state is the particular way in which it computes arc length.

To formalize this idea, we associate with each state $i$ the following quantities: (i) a feature-dependent matrix $g_i(\boldsymbol{x})$ that can be used to compute arc lengths, as in eq. (2); (ii) a decay parameter $\lambda_i$ that measures the probability per unit arc length that $s$ makes a transition from state $i$ to some other state; and (iii) a set of transition probabilities $a_{ij}$, where $a_{ij}$ represents the probability that—having decayed out of state $i$—the variable $s$ makes a transition to state $j$. Thus, $a_{ij}$ defines a stochastic transition matrix with zero elements along the diagonal and rows that sum to one: $a_{ii} = 0$ and $\sum_j a_{ij} = 1$. A Markov process is defined by the set of differential equations:

$$\frac{dp_i}{dt} = -\lambda_i p_i \left[ \dot{\boldsymbol{x}}^T g_i(\boldsymbol{x}) \dot{\boldsymbol{x}} \right]^{\frac{1}{2}} + \sum_{j \neq i} \lambda_j p_j a_{ji} \left[ \dot{\boldsymbol{x}}^T g_j(\boldsymbol{x}) \dot{\boldsymbol{x}} \right]^{\frac{1}{2}}, \tag{3}$$

where $p_i(t)$ denotes the (forward) probability that $s$ is in state $i$ at time $t$, based on its history up to that point in time. The right hand side of eq. (3) consists of two competing terms. The first term computes the probability that $s$ decays out of state $i$; the second computes the probability that $s$ decays into state $i$. Both terms are proportional to measures of arc length, making the evolution of $p_i$ along the curve of $\boldsymbol{x}$ invariant to nonlinear warpings of time. The decay parameter, $\lambda_i$, controls the typical amount of arc length traversed in state $i$; it may be viewed as

an inverse lifetime or—to be more precise—an inverse *lifelength*. The entire process is Markovian because the evolution of $p_i$ depends only on quantities available at time $t$.

## 2.3 Decoding

Given a trajectory $\boldsymbol{x}(t)$, the Markov process in eq. (3) gives rise to a conditional probability distribution over possible segmentations, $s(t)$. Consider the segmentation in which $s(t)$ takes the value $s_k$ between times $t_{k-1}$ and $t_k$, and let

$$\ell_{s_k} \;=\; \int_{t_{k-1}}^{t_k} dt \, \left[ \dot{\boldsymbol{x}}^T g_{s_k}(\boldsymbol{x}) \, \dot{\boldsymbol{x}} \right]^{\frac{1}{2}} \tag{4}$$

denote the arc length traversed in state $s_k$. By integrating eq. (3), one can show that the probability of remaining in state $s_k$ decays exponentially with the arc length $\ell_{s_k}$. Thus, the conditional probability of the overall segmentation is given by:

$$\Pr[s, \ell | \boldsymbol{x}] = \prod_{k=1}^{n} \lambda_{s_k} e^{-\lambda_{s_k} \ell_{s_k}} \prod_{k=0}^{n} a_{s_k s_{k+1}}, \tag{5}$$

where we have used $s_0$ and $s_{n+1}$ to denote the START and END states of the Markov process. The first product in eq. (5) multiplies the probabilities that each segment traverses exactly its observed arc length. The second product multiplies the probabilities for transitions between states $s_k$ and $s_{k+1}$. The leading factors of $\lambda_{s_k}$ are included to normalize each state's distribution over observed arc lengths.

There are many important quantities that can be computed from the distribution, $\Pr[s|\boldsymbol{x}]$. Of particular interest for ASR is the most probable segmentation: $s^*(\boldsymbol{x}) = \arg\max_{s,\ell} \left\{ \ln \Pr[s, \ell | \boldsymbol{x}] \right\}$. As described elsewhere (Saul, 1997), this maximization can be performed by discretizing the time axis and applying a dynamic programming procedure. The resulting algorithm is similar to the Viterbi procedure for maximum likelihood decoding (Rabiner & Juang, 1993).

## 2.4 Parameter estimation

The parameters $\{\lambda_i, a_{ij}, g_i(\boldsymbol{x})\}$ in MPCs are estimated from training data to maximize the log-likelihood of target segmentations. In our preliminary experiments with MPCs, we estimated only the metric parameters, $g_i(\boldsymbol{x})$; the others were assigned the default values $\lambda_i = 1$ and $a_{ij} = 1/f_i$, where $f_i$ is the fanout of state $i$. The metrics $g_i(\boldsymbol{x})$ were assumed to have the parameterized form:

$$g_i(\boldsymbol{x}) = \sigma_i^{-1} \Phi_i^2(\boldsymbol{x}), \tag{6}$$

where $\sigma_i$ is a positive definite matrix with unit determinant, and $\Phi_i(\boldsymbol{x})$ is a non-negative scalar-valued function of $\boldsymbol{x}$. For the experiments in this paper, the form of $\Phi_i(\boldsymbol{x})$ was fixed so that the MPCs reduced to HMMs as a special case, as described in the next section. Thus the only learning problem was to estimate the matrix parameters $\sigma_i$. This was done using the reestimation formula:

$$\sigma_i \;\leftarrow\; C \int dt \, \frac{\dot{\boldsymbol{x}}\dot{\boldsymbol{x}}^T}{[\dot{\boldsymbol{x}}^T \sigma_i^{-1} \dot{\boldsymbol{x}}]^{\frac{1}{2}}} \, \Phi_i(\boldsymbol{x}(t)), \tag{7}$$

where the integral is over all speech segments belonging to state $i$, and the constant $C$ is chosen to enforce the determinant constraint $|\sigma_i| = 1$. For fixed $\Phi_i(\boldsymbol{x})$, we have shown previously (Saul, 1997) that this iterative update leads to monotonic increases in the log-likelihood.

## 2.5   Relation to HMMs and previous work

There are several important differences between HMMs and MPCs. HMMs parameterize joint distributions of the form: $\Pr[s, x] = \prod_t \Pr[s_{t+1}|s_t] \Pr[x_t|s_t]$. Thus, in HMMs, parameter estimation is directed at learning a *synthesis* model, $\Pr[x|s]$, while in MPCs, it is directed at learning a *segmentation* model, $\Pr[s, \ell|x]$. The direction of conditioning on $x$ is a crucial difference. MPCs do not attempt to learn anything as ambitious as a joint distribution over acoustic feature trajectories.

HMMs and MPCs also differ in how they weight the speech signal. In HMMs, each state contributes an amount to the overall log-likelihood that grows in proportion to its duration in time. In MPCs, on the other hand, each state contributes an amount that grows in proportion to its arc length. Naturally, the weighting by arc length attaches a more important role to short-lived but *non-stationary* phonemes, such as consonants. It also guarantees the invariance to nonlinear warpings of time (to which the predictions of HMMs are quite sensitive).

In terms of previous work, our motivation for MPCs resembles that of Tishby (1990), who several years ago proposed a dynamical systems approach to speech processing. Because MPCs exploit the continuity of acoustic feature trajectories, they also bear some resemblance to so-called *segmental* HMMs (Ostendorf et al, 1996). MPCs nevertheless differ from segmental HMMs in two important respects: the invariance to nonlinear warpings of time, and the emphasis on learning a segmentation model $\Pr[s, \ell|x]$, as opposed to a synthesis model, $\Pr[x|s]$.

Finally, we note that admitting a slight generalization in the concept of arc length, we can essentially realize HMMs as a *special case* of MPCs. This is done by computing arc lengths along the *spacetime* trajectories $z(t) = \{x(t), t\}$—that is to say, replacing eq. (1) by $dL^2 = [\dot{z}^T g(z) \dot{z}]dt^2$, where $\dot{z} = \{\dot{x}, 1\}$ and $g(z)$ is a spacetime metric. This relaxes the invariance to nonlinear warpings of time and incorporates both movement in acoustic feature space *and* duration in time as measures of phonemic evolution. Moreover, in this setting, one can mimic the predictions of HMMs by setting the $\sigma_i$ matrices to have only one non-zero element (namely, the diagonal element for delta-time contributions to the arc length) and by defining the functions $\Phi_i(x)$ in terms of HMM emission probabilities $P(x|i)$ as:

$$\Phi_i(x) = -\ln\left[\frac{P(x|i)}{\sum_k P(x|k)}\right]. \tag{8}$$

This relation is important because it allows us to *initialize* the parameters of an MPC by those of a continuous-density HMM. This initialization was used in all the experiments reported below.

## 3   Automatic speech recognition

Both HMMs and MPCs were used to build connected speech recognizers. Training and test data came from speaker-independent databases of telephone speech. All data was digitized at the caller's local switch and transmitted in this form to the receiver. For feature extraction, input telephone signals (sampled at 8 kHz and band-limited between 100-3800 Hz) were pre-emphasized and blocked into 30ms frames with a frame shift of 10ms. Each frame was Hamming windowed, autocorrelated, and processed by LPC cepstral analysis to produce a vector of 12 liftered cepstral coefficients (Rabiner & Juang, 1993). The feature vector was then augmented by its normalized log energy value, as well as temporal derivatives of first and second order. Overall, each frame of speech was described by 39 features. These features were used differently by HMMs and MPCs, as described below.

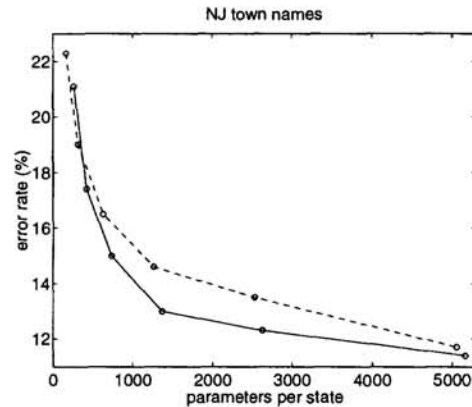

| Mixtures | HMM (%) | MPC (%) |
|:--------:|:-------:|:-------:|
| 2  | 22.3 | 20.9 |
| 4  | 18.9 | 17.5 |
| 8  | 16.5 | 15.1 |
| 16 | 14.6 | 13.3 |
| 32 | 13.5 | 12.3 |
| 64 | 11.7 | 11.4 |

Table 1: Word error rates for HMMs (dashed) and MPCs (solid) on the task of recognizing NJ town names. The table shows the error rates versus the number of mixture components; the graph, versus the number of parameters per hidden state.

Recognizers were evaluated on two tasks. The first task was recognizing New Jersey town names (e.g., Newark). The training data for this task (Sachs et al, 1994) consisted of 12100 short phrases, spoken in the seven major dialects of American English. These phrases, ranging from two to four words in length, were selected to provide maximum phonetic coverage. The test data consisted of 2426 isolated utterances of 1219 New Jersey town names and was collected from nearly 100 speakers. Note that the training and test data for this task have non-overlapping vocabularies.

Baseline recognizers were built using 43 left-to-right continuous-density HMMs, each corresponding to a context-independent English phone. Phones were modeled by three-state HMMs, with the exception of background noise, which was modeled by a single state. State emission probabilities were computed by mixtures of Gaussians with diagonal covariance matrices. Different sized models were trained using $M = 2$, 4, 8, 16, 32, and 64 mixture components per hidden state; for a particular model, the number of mixture components was the same across all states. Parameter estimation was handled by a Viterbi implementation of the Baum-Welch algorithm.

MPC recognizers were built using the same overall grammar. Each hidden state in the MPCs was assigned a metric $g_i(x) = \sigma_i^{-1}\Phi_i^2(x)$. The functions $\Phi_i(x)$ were initialized (and fixed) by the state emission probabilities of the HMMs, as given by eq. (8). The matrices $\sigma_i$ were estimated by iterating eq. (7). We computed arc lengths along the 14 dimensional spacetime trajectories through cepstra, log-energy, and time. Thus each $\sigma_i$ was a $14 \times 14$ symmetric matrix applied to tangent vectors consisting of delta-cepstra, delta-log-energy, and delta-time.

The table in figure 1 shows the results of these experiments comparing MPCs to HMMs. For various model sizes (as measured by the number of mixture components), we found the MPCs to yield consistently lower error rates than the HMMs. The graph in figure 1 plots these word error rates versus the number of modeling parameters per hidden state. This graph shows that the MPCs are not outperforming the HMMs merely because they have extra modeling parameters (i.e., the $\sigma_i$ matrices). The beam widths for the decoding procedures in these experiments were chosen so that corresponding recognizers activated roughly equal numbers of arcs.

The second task in our experiments involved the recognition of connected alpha-digits (e.g., N Z 3 V J 4 E 3 U 2). The training and test data consisted of

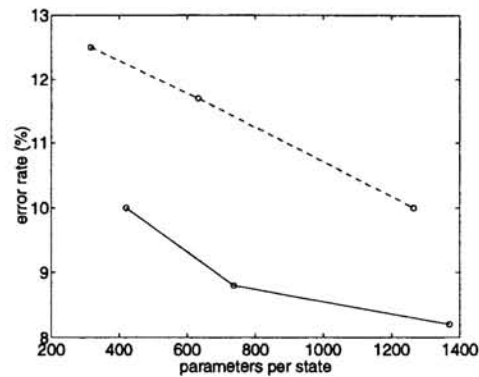

| Mixtures | HMM (%) | MPC (%) |
|:---:|:---:|:---:|
| 2 | 12.5 | 10.0 |
| 4 | 10.7 | 8.8 |
| 8 | 10.0 | 8.2 |

Figure 2: Word error rates for HMMs and MPCs on the task of recognizing connected alpha-digits. The table shows the error rates versus the number of mixture components; the graph, versus the number of parameters per hidden state.

14622 and 7255 utterances, respectively. Recognizers were built from 285 sub-word HMMs/MPCs, each corresponding to a context-dependent English phone. The recognizers were trained and evaluated in the same way as the previous task. Results are shown in figure 2.

While these results demonstrate the viability of MPCs for automatic speech recognition, several issues require further attention. The most important issues are feature selection—how to define meaningful acoustic trajectories from the raw speech signal—and learning—how to parameterize and estimate the hidden state metrics $g_i(x)$ from sampled trajectories $\{x(t)\}$. These issues and others will be studied in future work.

# References

M. P. do Carmo (1976). *Differential Geometry of Curves and Surfaces*. Prentice Hall.

M. Ostendorf, V. Digalakis, and O. Kimball (1996). From HMMs to segment models: a unified view of stochastic modeling for speech recognition. *IEEE Transactions on Acoustics, Speech and Signal Processing*, 4:360–378.

L. Rabiner and B. Juang (1993). *Fundamentals of Speech Recognition*. Prentice Hall, Englewood Cliffs, NJ.

R. Sachs, M. Tikijian, and E. Roskos (1994). United States English subword speech data. *AT&T unpublished report*.

L. Saul (1998). Automatic segmentation of continuous trajectories with invariance to nonlinear warpings of time. In *Proceedings of the Fifteenth International Conference on Machine Learning*, 506–514.

M. A. Siegler and R. M. Stern (1995). On the effects of speech rate in large vocabulary speech recognition systems. In *Proceedings of the 1995 IEEE International Conference on Acoustics, Speech, and Signal Processing*, 612–615.

N. Tishby (1990). A dynamical system approach to speech processing. In *Proceedings of the 1990 IEEE International Conference on Acoustics, Speech, and Signal Processing*, 365–368.
